# Affine Structure From Sound

**Sebastian Thrun**

Stanford AI Lab
Stanford University, Stanford, CA 94305
Email: thrun@stanford.edu

## Abstract

We consider the problem of localizing a set of microphones together with a set of external acoustic events (e.g., hand claps), emitted at unknown times and unknown locations. We propose a solution that approximates this problem under a far field approximation defined in the calculus of affine geometry, and that relies on singular value decomposition (SVD) to recover the affine structure of the problem. We then define low-dimensional optimization techniques for embedding the solution into Euclidean geometry, and further techniques for recovering the locations and emission times of the acoustic events. The approach is useful for the calibration of ad-hoc microphone arrays and sensor networks.

## 1   Introduction

Consider a set of acoustic sensors (microphones) for detecting acoustic events in the environment (e.g., a hand clap). The *structure from sound* (SFS) problem addresses the problem of simultaneously localizing a set of $N$ sensors and a set of $M$ external acoustic events, whose locations and emission times are unknown.

The SFS problem is relevant to the spatial calibration problem for microphone arrays. Classically, microphone arrays are mounted on fixed brackets of known dimensions; hence there is no spatial calibration problem. *Ad-hoc microphone arrays*, however, involve a person placing microphones at arbitrary locations with limited knowledge as to where they are. Today's best practice requires a person to measure the distance between the microphones by hand, and to apply algorithms such as multi-dimensional scaling (MDS) [1] for recovering their locations. When sensor networks are deployed from the air [4], manual calibration may not be an option. Some techniques rely on GPS receivers [8]. Others require a capability to emit and sense wireless radio signals [5] or sounds [9, 10], which are then used to estimate relative distances between microphones (directly or indirectly, as in [9]). Unfortunately, wireless signal strength is a poor estimator of range, and active acoustic and GPS localization techniques are uneconomical in that they consume energy and require additional hardware. In contrast, SFS relies on environmental acoustic events such as hand claps, which are *not* generated by the sensor network. The general SFS problem was previously treated in [2] under the name *passive localization*. A related paper [3] describes a technique for *incrementally* localizing a microphone relative to a well-calibrated microphone array through external sound events.

In this paper, the *structure from sound (SFS) problem* is defined as the simultaneous localization problem of $N$ sound sensors and $M$ acoustic events in the environment detected by these sensors. Each event occurs at an unknown time and an unknown location. The

sensors are able to measure the detection times of the event. We assume that the clocks of the sensors are synchronized (see [6]); that events are spaced sufficiently far apart in time to make the association between different sensors unambiguous; and we also assume absence of sound reverberation. For the ease of representation, the paper assumes a 2D world; although the technique is easily generalized to 3D.

Under the assumption of independent and identically distributed (iid) Gaussian noise, the SFS problem can be formulated as a least squares problem in a space over three types of variables: the locations of the microphones, the locations of the acoustic events, and their emission times. However, this least squares problem is plagued by local minima, and the number of constraints is quite large.

The gist of this paper transforms this optimization problem into a sequence of simpler problems, some of which can be solved optimally, without the danger of getting stuck in local minima. The key transformation involves a far field approximation, which presupposes that the sound sources are relatively far away from the sensors. This approximation reformulates the problem as one of recovering the incident angle of the acoustic signal, which is the same for all sensors for any fixed acoustic event. The resulting optimization problem is still non-linear; however, by relaxing the laws of Euclidean geometry into the more general calculus of affine geometry, the optimization problem can be solved by singular value decomposition (SVD). The resulting solution is mapped back into Euclidean space by optimizing a matrix of size $2 \times 2$, which is easily carried out using gradient descent. A subsequent non-linear optimization step overcomes the far field approximation and enables the algorithm to recover locations and emission times of the defining acoustic events. Experimental results illustrate that our approach reliably solves hard SFS problems where gradient-based techniques consistently fail.

Our approach is similar in spirit to the affine solution to the structure from motion (SFM) problem proposed by a seminal paper by Tomasi&Kanade [11], which was later extended to the non-orthographic case [7]. Like us, these authors expressed the structure finding problem using affine geometry, and applied SVD for solving it. SFM is of course defined for cameras, not for microphone arrays. Camera measure angles, whereas microphones measure range. This paper establishes an affine solution to the structure from sound problem that tends to work well in practice.

## 2 Problem Definition

### 2.1 Setup

We are given $N$ sensors (microphones) located in a 2D plane. We shall denote the location of the $i$-th sensor by $(x_i \ y_i)$, which defined the following sensor location matrix of size $N \times 2$:

$$X \quad = \quad \begin{pmatrix} x_1 & y_1 \\ x_2 & y_2 \\ \vdots & \vdots \\ x_N & y_N \end{pmatrix} \tag{1}$$

We assume that the sensor array detects $M$ acoustic events. Each event has as unknown coordinate and an unknown emission time. The coordinate of the $j$-th event shall be denoted $(a_j \ b_j)$, providing us with the event location matrix $A$ of size $M \times 2$. The emission time of the $j$-th acoustic event is denoted $t_j$, resulting in the vector $T$ of length $M$:

$$A \quad = \quad \begin{pmatrix} a_1 & b_1 \\ a_2 & b_2 \\ \vdots & \vdots \\ a_M & b_M \end{pmatrix} \qquad\qquad T \quad = \quad \begin{pmatrix} t_1 \\ t_2 \\ \vdots \\ t_M \end{pmatrix} \tag{2}$$

$X$, $A$, and $T$, comprise the set of unknown variables. In problems such as sensor calibration, only $X$ is of interest. In general SFS applications, $A$ and $T$ might also be of interest.

## 2.2 Measurement Data

In SFS, the variables $X$, $A$, and $T$ are recovered from data. The *data* establishes the detection times of the acoustic events by the individual sensors. Specifically, the *data matrix* is of the form:

$$D = \begin{pmatrix} d_{1,1} & d_{1,2} & \cdots & d_{1,M} \\ d_{2,1} & d_{2,2} & \cdots & d_{2,M} \\ \vdots & \vdots & \ddots & \vdots \\ d_{N,1} & d_{N,2} & \cdots & d_{N,M} \end{pmatrix} \tag{3}$$

Here each $d_{i,j}$ denotes the detection time of acoustical event $j$ by sensor $i$. Notice that we assume that there is no *data association problem.* Even if all acoustic events sound alike, the correspondence between different detections is easily established as long as there exists sufficiently long time gaps between any two sound events.

The matrix $D$ is a random field induced by the laws of sound propagation (without reverberation). In the absence of measurement noise, each $d_{i,j}$ is the sum of the corresponding emission time $t_j$, plus the time it takes for sound to travel from $(a_j \ b_j)$ to $(x_i \ y_i)$:

$$d_{i,j} = t_j + c^{-1} \left| \begin{pmatrix} x_i \\ y_i \end{pmatrix} - \begin{pmatrix} a_j \\ b_j \end{pmatrix} \right| \tag{4}$$

Here $| \cdot |$ denotes the L2 norm (Euclidean distance), and $c$ denoted the speed of sound.

## 2.3 Relative Formulation

Obviously, we cannot recover the global coordinates of the sensors. Hence, without loss of generality, we define the first sensor's location as $x_1 = y_1 = 0$. This gives us the *relative location matrix* for the sensors:

$$\bar{X} = \begin{pmatrix} x_2 - x_1 & y_2 - y_1 \\ x_3 - x_1 & y_3 - y_1 \\ \vdots & \vdots \\ x_N - x_1 & y_N - y_1 \end{pmatrix} \tag{5}$$

This relative sensor location matrix is of dimension $(N-1) \times 2$.

It shall prove convenient to subtract from the arrival time $d_{i,j}$ the arrival time $d_{1,j}$ measured by the first sensor $i = 1$. This *relative arrival time* is defined as $\Delta_{i,j} := d_{i,j} - d_{1,j}$. In the relative arrival time, the absolute emission times $t_j$ cancel out:

$$\Delta_{i,j} = t_j + c^{-1} \left| \begin{pmatrix} x_i \\ y_i \end{pmatrix} - \begin{pmatrix} a_j \\ b_j \end{pmatrix} \right| - t_j - c^{-1} \left| \begin{pmatrix} a_j \\ b_j \end{pmatrix} \right|$$

$$= c^{-1} \left\{ \left| \begin{pmatrix} x_i \\ y_i \end{pmatrix} - \begin{pmatrix} a_j \\ b_j \end{pmatrix} \right| - \left| \begin{pmatrix} a_j \\ b_j \end{pmatrix} \right| \right\} \tag{6}$$

We now define the matrix of relative arrival times:

$$\Delta = \begin{pmatrix} d_{2,1} - d_{1,1} & d_{2,2} - d_{1,2} & \cdots & d_{2,M} - d_{1,M} \\ d_{3,1} - d_{1,1} & d_{3,2} - d_{1,2} & \cdots & d_{3,M} - d_{1,M} \\ \vdots & \vdots & \ddots & \vdots \\ d_{N,1} - d_{1,1} & d_{N,2} - d_{1,2} & \cdots & d_{N,M} - d_{1,M} \end{pmatrix} \tag{7}$$

This matrix $\Delta$ is of dimension $(N-1) \times M$.

## 2.4 Least Squares Formulation

The relative sensor locations $X$ and the corresponding locations of the acoustic events $A$ can now be recovered through the following least squares problem. This optimization seeks to identify $X$ and $A$ so as to minimize the quadratic difference between the predicted relative measurements and the actual measurements.

$$\langle A^*, X^* \rangle = \underset{X,A}{\operatorname{argmin}} \sum_{i=2}^{N} \sum_{j=1}^{M} \left\{ \left| \begin{pmatrix} x_i \\ y_i \end{pmatrix} - \begin{pmatrix} a_j \\ b_j \end{pmatrix} \right| - \left| \begin{pmatrix} a_j \\ b_j \end{pmatrix} \right| - \Delta_{i,j} \right\}^2 \tag{8}$$

The minimum of this expression is a maximum likelihood solution for the SFS problem under the assumption of iid Gaussian measurement noise.

If emission times are of interest, they are now easily recovered by the following weighted mean:

$$T^* \quad = \quad \frac{1}{N} \sum_{i=1}^{N} d_{i,j} - c \left| \begin{pmatrix} x_i \\ y_i \end{pmatrix} - \begin{pmatrix} a_j \\ b_j \end{pmatrix} \right| \tag{9}$$

The minimum of Eq. 8 is not unique. This is because any solution can be rotated around the origin of the coordinate system, and mirrored through any axis intersecting the origin. This shall not concern us, as we shall be content with *any* solution of Eq. 8; others are then easily generated.

What is of concern, however, is the fact that minimizing Eq. 8 is difficult. A straw man algorithm—which tends to work poorly in practice—involves starting with random guesses for $X$ and $A$ and then adjusting them in the direction of the negative gradient until convergence. As we shall show experimentally, such gradient algorithms work poorly in practice because of the large number of local minima.

## 3   The Far Field Approximation

The essence of our approximation pertains to the fact that for far range acoustic events—i.e., events that are (infinitely) far away from the sensor array—the incoming sound wave hits each sensor at the same incident angle. Put differently, the rays connecting the location of an acoustic event $(a_j \ b_j)$ with each of the perceiving sensors $(x_i \ y_i)$ are approximately parallel for all $i$ (but *not* for all $j$!). Under the far field approximation, these incident angles are entirely parallel. Thus, all that matters are the incident angle of the acoustic events.

To derive an equation for this case, it shall prove convenient to write the Euclidean distance between a sensor and an acoustic event as a function of the incident angle $\alpha$. This angle is given by the four-quadrant extension of the arctan function:

$$\alpha_{i,j} \quad = \quad \text{arctan2} \, \frac{b_j - y_i}{a_j - x_i} \tag{10}$$

The Euclidean distance between $(a_j \ b_j)$ and $(x_i \ y_i)$ can now be written as

$$\left| \begin{pmatrix} x_i \\ y_i \end{pmatrix} - \begin{pmatrix} a_j \\ b_j \end{pmatrix} \right| \quad = \quad (\cos \alpha_{i,j} \ \ \sin \alpha_{i,j}) \begin{pmatrix} a_j - x_i \\ b_j - y_i \end{pmatrix} \tag{11}$$

For far-away points $(a_j \ b_j)$, we can safely assume that all incident angles for the $j$-th acoustic event are identical:

$$\alpha_j \quad := \quad \alpha_{1,j} \ = \ \alpha_{2,j} \ = \ \dots \ = \ \alpha_{N,j} \tag{12}$$

Hence we substitute $\alpha_j$ for $\alpha_{i,j}$ in Eq. 11. Plugging this back into Eq. 6, this gives us the following expression for $\Delta_{i,j}$:

$$\begin{aligned} \Delta_{i,j} \quad &= \quad c^{-1} \left\{ \left| \begin{pmatrix} x_i \\ y_i \end{pmatrix} - \begin{pmatrix} a_j \\ b_j \end{pmatrix} \right| - \left| \begin{pmatrix} a_j \\ b_j \end{pmatrix} \right| \right\} \\ &\approx \quad c^{-1} \left\{ (\cos \alpha_j \ \ \sin \alpha_j) \left[ \begin{pmatrix} a_j - x_i \\ b_j - y_i \end{pmatrix} - \begin{pmatrix} a_j \\ b_j \end{pmatrix} \right] \right\} \\ &= \quad c^{-1} (\cos \alpha_j \ \ \sin \alpha_j) \begin{pmatrix} x_i \\ y_i \end{pmatrix} \end{aligned} \tag{13}$$

This leads to the following non-linear least squares problem for the desired sensor locations:

$$\langle X^*, \alpha_1^*, \dots, \alpha_M^* \rangle \quad = \quad \underset{X, \alpha_1, \dots, \alpha_M}{\text{argmin}} \left| X \begin{pmatrix} \cos \alpha_1 & \cos \alpha_2 & \cdots & \cos \alpha_M \\ \sin \alpha_1 & \sin \alpha_2 & \cdots & \sin \alpha_M \end{pmatrix} - \Delta \right|^2 \tag{14}$$

The reader many notice that in this formulation of the SFS problem, the locations of the sound events $(a_j, b_j)$ have been replaced by $\alpha_j$, the incident angles of the sound waves.

One might think of this as the "ortho-acoustic" model of sound propagation (in analogy to the orthographic camera model in computer vision). The ortho-acoustic projection reduces the number of variables in the optimization. However, the argument in the quadratic expression is still non-linear, due to the non-linear trigonometric functions involved.

## 4 Affine Solution for the Sensor Locations

Eq. 14 is trivially solvable in the space of *affine geometry*. Following [11], in affine geometry projections can be arbitrary linear functions, not just rotations and translations. Specifically, let us replace the specialized matrix

$$\begin{pmatrix} \cos\alpha_1 & \cos\alpha_2 & \cdots & \cos\alpha_M \\ \sin\alpha_1 & \sin\alpha_2 & \cdots & \sin\alpha_M \end{pmatrix} \tag{15}$$

by a general $2 \times M$ matrix of the form

$$\Gamma = \begin{pmatrix} \gamma_{1,1} & \gamma_{1,2} & \cdots & \gamma_{1,M} \\ \gamma_{2,1} & \gamma_{2,2} & \cdots & \gamma_{2,M} \end{pmatrix} \tag{16}$$

This leads to the least squares problem

$$\langle X^*, \Gamma^* \rangle = \underset{X,\Gamma}{\operatorname{argmin}} |X\Gamma - \Delta|^2 \tag{17}$$

In the noise free-case case, we know that there must exist a $X$ and a $\Gamma$ for which $X\Gamma = \Delta$. This suggests that the rank of $\Delta$ should be 2, since it is the product of a matrix of size $(N-1) \times 2$ and a matrix of size $2 \times M$.

Further, we can recover both $X$ and $\Gamma$ via singular value decomposition (SVD). Specifically, we know that the matrix $\Delta$ can be decomposed as into three other matrices, $U$, $V$, and $W$:

$$UVW^T = \operatorname{svd}(\Delta) \tag{18}$$

where $U$ is a matrix of size $(N-1) \times 2$, $V$ a diagonal matrix of eigenvalues of size $2 \times 2$, and $W$ a matrix of size $M \times 2$. In practice, $\Delta$ might be of higher rank because of noise or because of violations of the far field assumption, but it suffices to restrict the consideration to the first two eigenvalues.

The decomposition in Eq. 18 leads to the optimal affine solution of the SFS problem:

$$X = UV \quad \text{and} \quad \Gamma = W^T \tag{19}$$

However, this solution is not yet Euclidean, since $\Gamma$ might not be of the form of Eq. 15. Specifically, Eq. 15 is a function of angles, and each row in Eq. 15 must be of the form $\cos^2\gamma_j + \sin^2\gamma_j = 1$. Clearly, this constraint is not enforced in the SVD.

However, there is an easy "trick" for recovering a $X$ and $\Gamma$ for which this constraint is at least approximately met. The key insight is that for any invertible $2 \times 2$ matrix $C$,

$$X' = UVC^{-1} \quad \text{and} \quad \Gamma' = CW^T \tag{20}$$

is equally a solution to the factorization problem in Eq.18. This is because

$$X'\Gamma' = UVC^{-1}CW^T = UVW^T = X\Gamma \tag{21}$$

The remaining search problem, thus, is the problem of finding an appropriate matrix $C$ for which $\Gamma'$ is of the form of Eq. 15. This is a *non-linear* optimization problem, but it is much lower-dimensional than the original SFS problem (it only involves 4 parameters!).

Specifically, we seek a $C$ for which $\Gamma' = CW^T$ minimizes

$$C^* = \underset{C}{\operatorname{argmin}} \left| \underbrace{(1\ 1)\,(\Gamma' \cdot \Gamma')}_{(*)} - (1\ 1\ \cdots\ 1) \right|^2 \tag{22}$$

Here "·" denotes the dot product. The expression labeled $(*)$ evaluates to a vector of expressions of the form

$$(\gamma_{1,1}^2 + \gamma_{2,1}^2 \quad \gamma_{1,2}^2 + \gamma_{2,2}^2 \quad \cdots \quad \gamma_{1,M}^2 + \gamma_{2,M}^2) \tag{23}$$

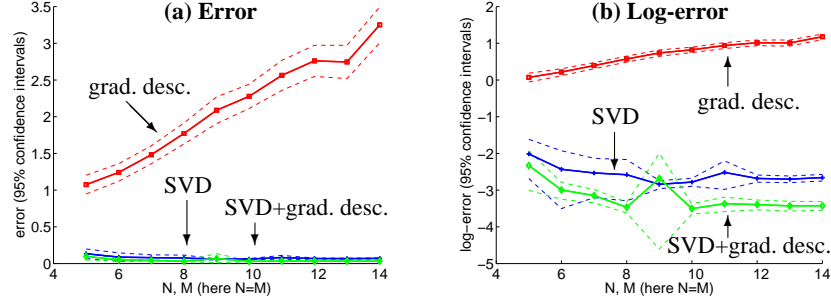

**Figure 1**: (a) Error and (b) log error for three different algorithms: gradient descent (red), SVD (blue), and SVD followed by gradient descent (green). Performance is shown for different values of $N$ and $M$, with $N = M$. The plot also shows 95% confidence bars.

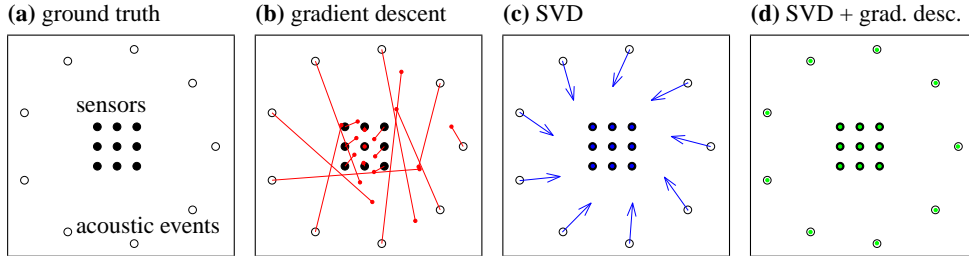

**(a)** ground truth   **(b)** gradient descent   **(c)** SVD   **(d)** SVD + grad. desc.

**Figure 2**: Typical SFS results for a simulated array of nine microphones spaced in a regular grid, surrounded by 9 sounds arranged on a circle. (a) Ground truth; (b) Result of plain gradient descent after convergence; the dashed lines visualize the residual error; (c) Result of the SVD with sound directions as indicated; and (d) Result of gradient descent initialized with our SVD result.

The minimization in Eq. 22 is carried out through standard gradient descent. It involves only 4 variables ($C$ is of the size $2 \times 2$), and each single iteration is linear in $O(N + M)$ (instead of the $O(NM)$ constraints that define Eq. 8). In (tens of thousands of) experiments with synthetic noise-free data, we find empirically that gradient descent reliably converges to the globally optimal solution.

## 5 Recovering the Acoustic Event Locations and Emission Times

With regards to the acoustic events, the optimization for the far field case only yields the incident angles. In the near field setting, in which the incident angles tend to differ for different sensors, it may be desirable to recover the locations $A$ of the acoustic event and the corresponding emission times $T$.

To determine these variables, we use the vector $X^*$ from the far field case as mere starting points in a subsequent gradient search. The event location matrix $A$ is initialized by selecting points sufficiently far away along the estimated incident angle for the far field approximation to be sound:

$$A = k\,\Gamma'^* \tag{24}$$

Here $\Gamma'^* = C^* W^T$ with $C^*$ defined in Eq. 22, and $k$ is a multiple of the diameter of the locations in $X$. With this initial guess for $A$, we apply gradient descent to optimize Eq. 8, and finally use Eq. 9 to recover $T$.

## 6 Experimental Results

We ran a series of simulation experiments to characterize the quality of our algorithm, especially in comparison with the obvious nonlinear least squares problem (Eq. 8) from which it is derived. Fig. 1 graphs the residual error as a function of the number of sensors

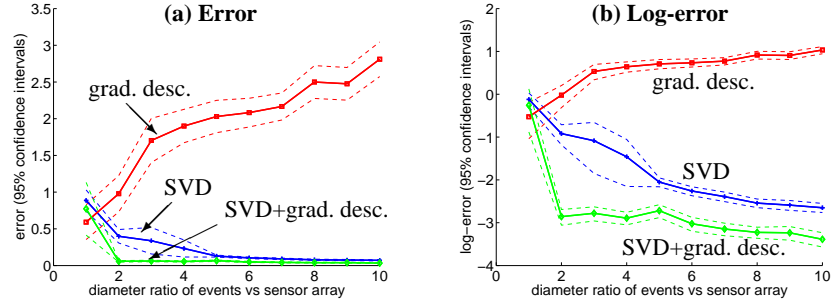

**Figure 3**: (a) Error and (b) log-error for three different algorithms (gradient descent in red, SVD in blue, and SVD followed by gradient descent in green), graphed here for varying distances of the sound events to the sensor array. An error above 2 means the reconstruction has entirely failed. All diagrams also show the 95% confidence intervals, and we set $N = M = 10$.

**(a)** One of our motes
used to generate the data

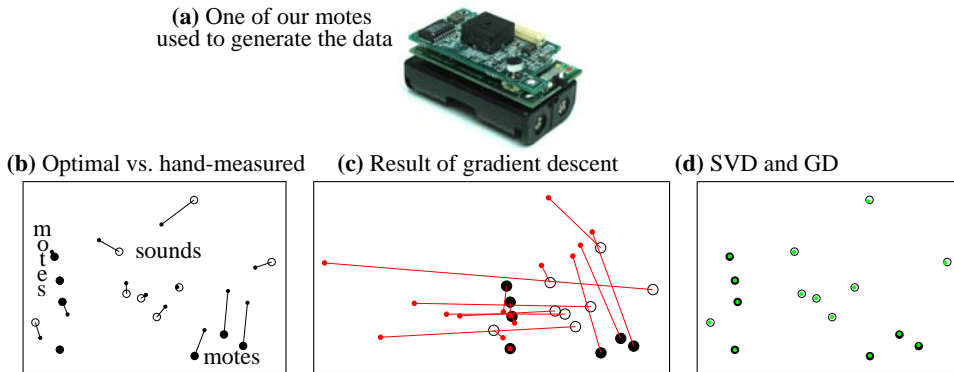

**(b)** Optimal vs. hand-measured  **(c)** Result of gradient descent  **(d)** SVD and GD

**Figure 4**: Results using our seven sensor motes as the sensor array, and a seventh mote to generate sound events. (a) A mote; (b) the globally optimal solution (big circles) compared to the hand-measures locations (small circles); (c) a typical result of vanilla gradient descent; and (d) the result of our approach, all compared to the optimal solution given the (noisy) data.

$N$ and acoustic events $M$ (here $N = M$). Panel (a) plots the regular error along with 95% confidence intervals, and panel (b) the corresponding log-error. Clearly, as $N$ and $M$ increase, plain gradient descent tends to diverge, whereas our approach converges. Each data point in these graphs was obtained by averaging 1,000 random configurations, in which sensors were sampled uniformly within an interval of $1\times1$m; sounds were placed at varying ranges, from 2m to 10m. An example outcome (for a non-random configuration!) is shown in Fig. 2. This figure plots (a) a simulated sensor array consisting of 9 sensors with 9 sound sources arranged in a circle; and (b)-(d) the resulting reconstructions of our three methods. For the SVD result shown in (c), only the directions of the incoming sounds are shown.

An interesting question pertains to the effect of the far field approximation in cases where it is clearly violated. To examine the robustness of our approach, we ran a series of experiments in which we varied the diameter of the acoustic events relative to the diameter of the sensors. If this parameter is 1, the acoustic events are emitted in the same region as the microphones; for values such as 10, the events are far away.

Fig. 3 graphs the residual errors and log-errors. The further away the acoustic events, the better our results. However, even for nearby events, for which the far field assumption is clearly invalid, our approach generates results that are no worse than those of the plain gradient descent technique.

We also implemented our approach using a physical sensor array. Fig. 4 plots empirical results using a microphone array comprised of seven Crossbow sensor motes, one of which

is shown in Panel (a). Panels (b-d) compare the recovered structure with the one that globally minimizes the LMS error, which we obtain by running gradient descent using the hand-measured locations as starting point. Panel (a) in Fig. 4 shows the manually measured locations; the relatively high deviation to the LMS optimum is the result of measurement error, which is amplified by the fact that our motes are only spaced a few tens of centimeters apart from each other (the standard deviation in the timing error corresponds to a distance of 6.99cm, and the motes are placed between 14cm and 125cm apart). Panel (b) in Fig. 4 shows the solution of plain gradient descent applied to applied to Eq.8 and compares it to the optimal reconstruction; and Panel (c) illustrates our solution. In all plots the lines indicate residual error. This result shows that our method may work well on real-world data that is noisy and that does not adhere to the far field assumption.

## 7 Discussion

This paper considered the *structure from sound* problem and presented an algorithm for solving it. Our approach makes is possible to simultaneously recover the location of a collection of microphones, the locations of external acoustic events detected by these microphones, and the emission times for these events. By resorting to affine geometry, our approach overcomes the problem of local minima in the structure from sound problem.

There remain a number of open research issues. We believe the extension to 3-D is mathematically straightforward but requires empirical validation. The current approach also fails to address reverberation problems that are common in confined space. It shall further be interesting to investigate data association problems in the SFS framework, and to develop parallel algorithms that can be implemented on sensor networks with limited communication resources. Finally, of great interest should be the incomplete data case in which individual sensors may fail to detect acoustic events—a problem studied in [2].

## Acknowledgement

The motes data was made available by Rahul Biswas, which is gratefully acknowledged. We also acknowledge invaluable suggestions by three anonymous reviewers.

## References

[1] S.T. Birchfield and A. Subramanya. Microphone array position calibration by basis-point classical multidimensional scaling. *IEEE Trans. Speech and Audio Processing*, forthcoming.

[2] R. Biswas and S. Thrun. A passive approach to sensor network localization. IROS-04.

[3] J.C. Chen, R.E. Hudson, and K. Yao. Maximum likelihod source localization and unknown sensor location estimation for wideband signals in the near-field. *IEEE Trans. Signal Processing*, 50, 2002.

[4] P. Corke, S. Hrabar, R. Peterson, D. Rus, S. Saripalli, and G. Sukhatme. Deployment and connectivity repair of a sensor net with a flying robot. ISER-04.

[5] E. Elnahrawy, X. Li, and R. Martin. The limits of localization using signal strength: A comparative study. SECON-04.

[6] J. Elson and K. Romer. Wireless sensor networks: A new regime for time synchronization. HotNets-02.

[7] S. Mahamud and M. Hebert. Iterative projective reconstruction from multiple views. CVPR-00.

[8] D. Niculescu and B. Nath. Ad hoc positioning system (APS). GLOBECOM-01.

[9] V.C. Raykar, I.V. Kozintsev, and R. Lienhart. Position calibration of microphones and loudspeakers in distributed computing platforms. *IEEE transaction on Speech and Audio Processing*, 13(1), 2005.

[10] J. Sallai, G. Balogh, M. Maroti, and A. Ledeczi. Acoustic ranging in resource-constrained sensor networks. eCOTS-04.

[11] C. Tomasi and T. Kanade. Shape and motion from image streams under orthography: A factorization method. *IJCV*, 9(2), 1992.

[12] T.L. Tung, K. Yao, D. Chen, R.E. Hudson, and C.W. Reed. Source localization and spatial filtering using wideband music and maxiumum power beam forming for multimedia applications. In SIPS-99.
